# A VLSI Implementation of the Adaptive Exponential Integrate-and-Fire Neuron Model

**Sebastian Millner, Andreas Grübl, Karlheinz Meier,**
**Johannes Schemmel and Marc-Olivier Schwartz**
Kirchhoff-Institut für Physik
Ruprecht-Karls-Universität Heidelberg
`smillner@kip.uni-heidelberg.de`

## Abstract

We describe an accelerated hardware neuron being capable of emulating the adaptive exponential integrate-and-fire neuron model. Firing patterns of the membrane stimulated by a step current are analyzed in transistor level simulations and in silicon on a prototype chip. The neuron is destined to be the hardware neuron of a highly integrated wafer-scale system reaching out for new computational paradigms and opening new experimentation possibilities. As the neuron is dedicated as a universal device for neuroscientific experiments, the focus lays on parameterizability and reproduction of the analytical model.

## 1   Introduction

Since the beginning of neuromorphic engineering [1, 2] designers have had great success in building VLSI[1] neurons mimicking the behavior of biological neurons using analog circuits [3–8]. The design approaches are quite different though, as the desired functions constrain the design.

It has been argued [4] whether it is best to emulate an established model or to create a new one using analog circuits. The second way is gone by [3–7] for instance, aiming at the low power consumption and fault tolerance of neural computation to be used in a computational device in robotics for example. This can be done most effectively by the technology-driven design of a new model, fitted directly to biological results. We approach gaining access to the computational power of neural systems and creating a device being able to emulate biologically relevant spiking neural networks that can be reproduced in a traditional simulation environment for modeling. The use of a commonly known model enables modelers to do experiments on neuromorphic hardware and compare them to simulations. This design methodology has been applied successfully in [8, 9], implementing the conductance-based integrate-and-fire model [10]. The software framework `PyNN` [11, 12] even allows for directly switching between a simulator and the neuromorphic hardware device, allowing modelers to access the hardware on a high level without knowing all implementation details.

The hardware neuron presented here can emulate the adaptive exponential integrate-and-fire neuron model (AdEx) [13], developed within the FACETS-project [14]. The AdEx model can produce complex firing patterns observed in biology [15], like spike-frequency-adaptation, bursting, regular spiking, irregular spiking and transient spiking by tuning a limited number of parameters [16].

Completed by the reset conditions, the model can be described by the following two differential equations for the membrane voltage $V$ and the adaptation variable $w$:

$$-C_{\mathrm{m}}\frac{dV}{dt} = g_{\mathrm{l}}(V - E_1) - g_{\mathrm{l}}\Delta_{\mathrm{t}}e^{\left(\frac{V - V_{\mathrm{t}}}{\Delta_{\mathrm{t}}}\right)} + g_{\mathrm{e}}(t)(V - E_{\mathrm{e}}) + g_{\mathrm{i}}(t)(V - E_{\mathrm{i}}) + w; \qquad (1)$$

$$-\tau_w \frac{dw}{dt} = w - a(V - E_{\mathrm{l}}). \qquad (2)$$

$C_{\mathrm{m}}$, $g_{\mathrm{l}}$, $g_{\mathrm{e}}$ and $g_{\mathrm{i}}$ are the membrane capacitance, the leakage conductance and the conductances for excitatory and inhibitory synaptic inputs, where $g_{\mathrm{e}}$ and $g_{\mathrm{i}}$ depend on time and the inputs from other neurons. $E_{\mathrm{l}}$, $E_{\mathrm{i}}$ and $E_{\mathrm{e}}$ are the leakage reversal potential and the synaptic reversal potentials. The parameters $V_{\mathrm{t}}$ and $\Delta_{\mathrm{t}}$ are the effective threshold potential and the threshold slope factor. The time constant of the adaptation variable is $\tau_w$ and $a$ is called adaptation parameter. It has the dimension of a conductance.

If the membrane voltage crosses a certain threshold voltage $\Theta$, the neuron is reset:

$$V \to V_{\mathrm{reset}}; \qquad (3)$$

$$w \to w + b. \qquad (4)$$

The parameter $b$ is responsible for spike-triggered adaptation. Due to the sharp rise, created by the exponential term in equation 1, the exact value of $\Theta$ is not critical for the determination of the moment of a spike [13].

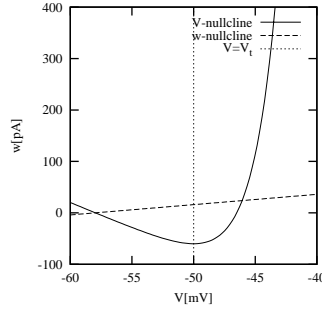

Figure 1: Phase plane of the AdEx model with parameters according to figure 4 d) from [16], stimulus excluded. $V$ and $w$ will be rising below their nullclines and falling above.

Figure 1 shows the phase plane of the AdEx model with its nullclines. The nullcline of a variable is the cline, where its time derivative is zero. The crossing off the nullclines in the left is the stable fix-point, where the trajectory is located in rest. A constant current stimulus will lift the $V$-nullcline. For $V > V_t$ below the $V$-nullcline, the derivative of $V$ is proportional to $V$ - the exponential dominates and V diverges until $\Theta$ is reached.

The neuron is integrated on a prototype chip called HICANN[2] [17–19] (figure 2) which has been produced in 2009. Each HICANN contains 512 dendrite membrane (DenMem) circuits (figure 3), each being connected to 224 dynamic input synapses. Neurons are built of DenMems by shorting their membrane capacitances gaining up to 14336 input synapses for a single neuron. The HICANN is prepared for integration in the FACETS wafer-scale system [17–19] allowing to interconnect 384 HICANNs on an uncut silicon wafer via a high speed bus system, so networks of up to 196 608 neurons can be emulated on a single wafer.

A major feature of the described hardware neuron is that the size of components allows working with an acceleration factor of $10^3$ up to $10^5$ compared to biological real time, enabling the operator to do several runs of an experiment in a short time to do large parameter sweeps and gain better statistics. Effects occurring on a longer timescale like long term synaptic plasticity could be emulated. This way the wafer-scale system can emerge as an alternative and an enhancement to traditional computer simulations in neuroscience. Another VLSI neuron designed with a time scaling factor is presented in [7]. This implementation is capable of reproducing lots of different firing patterns of cortical neurons, but has no direct correspondence to a neuron from the modeling area.

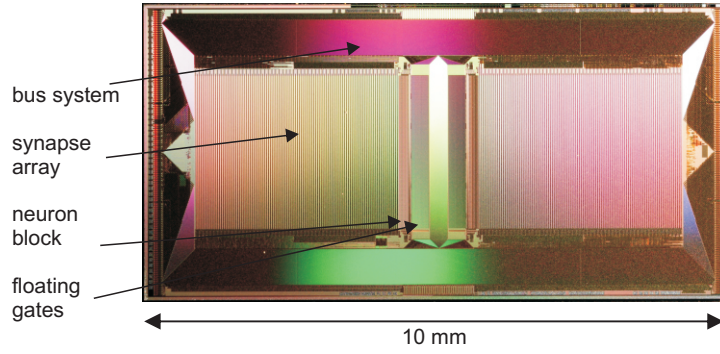

bus system

synapse array

neuron block

floating gates

10 mm

Figure 2: Photograph of the `HICANN`-chip

## 2 Neuron implementation

### 2.1 Neuron

The smallest part of a neuron is a `DenMem`, which implements the terms of the AdEx neuron described above. Each term is constructed by a single circuit using operational amplifiers (`OP`) and operational transconductance amplifiers (`OTA`) and can be switched off separately, so less complex models like the leaky integrate-and-fire model implemented in [9] can be emulated. `OTA`s directly model conductances for small input differences. The conductance is proportional to a biasing current. A first, not completely implemented version of the neuron has been proposed in [17]. Some simulation results of the actual neuron can be found in [19].

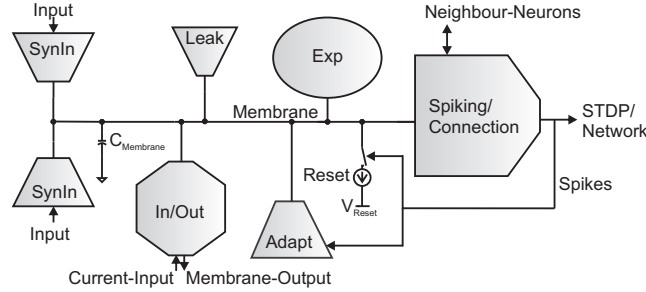

Figure 3: Schematic diagram of AdEx neuron circuit

Figure 3 shows a block diagram of a `DenMem`. During normal operation, the neuron gets rectangular shaped current pulses as input from the `synapse array` (figure 2) at one of the two synaptic input circuits. Inside these circuits the current is integrated by a leaky integrator OP-circuit resulting in a voltage that is transformed to a current by an `OTA`. Using this current as bias for another `OTA`, a sharply rising and exponentially decaying synaptic input conductance is created. Each `DenMem` is equipped with two synaptic input circuits, each having its own connection to the `synapse array`. The output of a synapse can be chosen between them, which allows for two independent synaptic channels which could be inhibitory or excitatory.

The leakage term of equation 1 can be implemented directly using an `OTA`, building a conductance between the leakage potential $E_l$ and the membrane voltage $V$.

Replacing the adaptation variable $w$ in equation 2 by $a(V_{\mathrm{adapt}} - E_l)$, results in:

$$-\tau_w \frac{dV_{\mathrm{adapt}}}{dt} = V_{\mathrm{adapt}} - V. \tag{5}$$

Now the time constant $\tau_w$ shall be created by a capacitance $C_{\text{adapt}}$ and a conductance $g_{\text{adapt}}$ and we get:

$$-C_{\text{adapt}} \frac{dV_{\text{adapt}}}{dt} = g_{\text{adapt}}(V_{\text{adapt}} - V).$$

(6)

We need to transform $b$ into a voltage using the conductance $a$ and get

$$C_{\text{adapt}} I_b t_{\text{pulse}} = \frac{b}{a}$$

(7)

where the fixed $t_{\text{pulse}}$ is the time a current $I_b$ increases $V_{\text{adapt}}$ on $C_{\text{adapt}}$ at each detected spike of a neuron. These resulting equations for adaptation can be directly implemented as a circuit.

A MOSFET[3] connected as a diode is used to emulate the exponential positive feedback of equation 1 (figure 4). To generate the correct gate source voltage, a non inverting amplifier multiplies the difference between the membrane voltage and a voltage $V_t$ by an adjustable factor. A simplified version of the circuit can be seen in figure 4. The gate source voltage of M1 is :

$$V_{\text{GSM1}} = \frac{R_1}{R_2}(V - V_{\text{t}})$$

(8)

Deployed in the equation for a MOSFET in sub-threshold mode this results in a current depending exponentially on $V$ following equation 1 where $\Delta_{\text{t}}$ can be adjusted via the resistors $R_1$ and $R_2$. The factor in front of the exponential $g_{\text{l}}\Delta_{\text{t}}$ and $V_{\text{t}}$ of the model can be changed by moving the circuits $V_{\text{t}}$. To realize huge (hundreds of $k\Omega$) variable resistors, the slope of the output characteristic of a MOSFET biased in saturation is used as replacement for $R_1$.

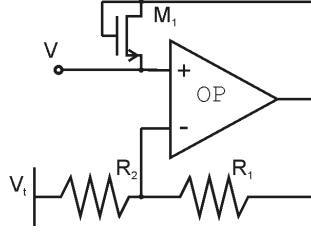

Figure 4: Simplified schematic of the exponential circuit

Our neuron detects a spike at a directly adjustable threshold voltage $\Theta$ - this is especially necessary as the circuit cannot only implement the AdEx model, but also less complex models. In a model without a sharp spike, like the one created by the positive feedback of the exponential term, spike timing very much depends on the exact voltage $\Theta$.

A detected spike triggers reseting of the membrane by a current pulse to a potential $V_{\text{reset}}$ for an adjustable time. Therefore our circuit supports basic modeling of a refractory period additionally to the modeling by the adaption variable.

## 2.2 Parameterization

In contrast to most other systems, we are using analog floating gate memories similar to [20] as storage device for the analog parameters of a neuron. Due to the small size of these cells, we are capable of providing most parameters individually for a single `DenMem` circuit. This way, matching issues can be counterbalanced, and different types of neurons can be implemented on a single chip enhancing the universality of the wafer-scale system.

Table 1 shows the parameters used in the implemented AdEx model and the parameter ranges aimed during design. Technical biasing parameters and parameters of the synaptic input circuits are excluded. Parameter ranges of several orders of magnitude are necessary, as our neurons can work in different time scalings relative to real time. This is achieved by switching between different multiplication factors for biasing currents. As these switches are parameterized globally, ranges of a parameter of a neuron group(one quarter of a `HICANN`) need to be in the same order of magnitude.

Table 1: Neuron parameters

| PARAMETER | SHARING | RANGE |
|---|---|---|
| $g_l$ | individual | 34 nS..4 μS |
| $a$ | individual | 34 nS..4 μS |
| $g_{adapt}$ | individual | 5 nS..2 μS |
| $I_b$ | individual | 200 nA..5 μA |
| $t_{pulse}$ | fixed | 18 ns |
| $V_{reset}$ | global | 0 V..1.8 V |
| $V_{exp}$ | individual | 0 V..1.8 V |
| $t_{reset}$ | global | 25 ns..500 ns |
| $C_{mem}$ | global | 400 fF or 2 pF |
| $C_{adapt}$ | fixed | 2 pF |
| $\Delta_t$ | individual | ..10 mV.. |
| $\Theta$ | individual | 0 V..1.8 V |

As a starting point for for the parameter ranges, [13] and [21] have been used. The chosen ranges allow leakage time constants $\tau_{mem} = C_{mem}/g_l$ at an acceleration factor of $10^4$ between 1 ms and 588 ms and an adaptation time constant $\tau_w$ between 10 ms and 5 s in terms of biological real time. So the parameters used in [22] are easily reached for instance. Switching to other acceleration modes, the regime for a biologically realistic operation is reduced as the needed time constants are shifted one order of magnitude.

As `OTAs` are used for modeling conductances, and linear operation for this type of devices can only be achieved for smaller voltage differences, it is necessary to limit the operating range of the variables $V$ and $V_{adapt}$ to some hundreds of millivolts. If this area is left, the `OTAs` will not work as a conductance anymore, but as a constant current, hence there will not be any more spike triggered adaptation for example.

A neuron can be composed of up to 64 `DenMem` circuit hence several different adaptation variables with different time constants for each are allowed.

## 2.3 Parameter mapping

For a given set of parameters from the AdEx model, we want to reproduce the exact same behavior with our hardware neuron. Therefore, a simple two-steps procedure was developed to translate biological parameters from the AdEx model to hardware parameters. The translation procedure is summarized in figure 5:

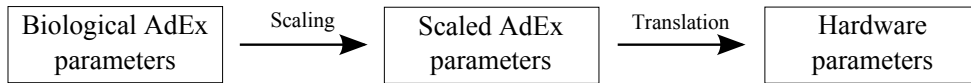

Figure 5: Biology to hardware parameter translation

The first step is to scale the biological AdEx parameters in terms of time and voltage. At this stage, the desired time acceleration factor is chosen, and applied to the two time constants of the model. Then, a voltage scaling factor is defined, by which the biological voltages parameters are multiplied. This factor has to be high enough to improve the signal-to-noise ratio in the hardware, but not too high to stay in the operating range of the `OTAs`.

The second step is to translate the parameters from the scaled AdEx model to hardware parameters. For this purpose, each part of the `DenMem` circuit was characterized in transistor-level simulations using a circuit simulator. This theoretical characterization was then used to establish mathematical relations between scaled AdEx parameters and hardware parameters.

## 2.4 Measurement capabilities

For neuron measuring purposes, the membrane can be either stimulated by incoming events from the `synapse array` - as an additional feature a Poisson event source is implemented on the chip - or by a programmable current. This current can be programmed up to few μA replaying 129 10 bit values using a sequencer and a digital-to-analog converter. Four current sources are implemented on the chip allowing to stimulate adjacent neurons individually. Currently, the maximum period of a current stimulus is limited to 33 μs, but this can be easily enhanced as the HICANN host interface allows an update of the value storage in real time.

The membrane voltage and all stored parameters in the floating gates can directly be measured via one of the two analog outputs of the `HICANN` chip. Membrane voltages of two arbitrary neurons can be read out at the same time.

To characterize the chip, parameters like the membrane capacitance need to be measured indirectly using the `OTA`, emulating $g_l$, as a current source example.

## 3 Results

Different firing patterns have been reproduced using our hardware neuron and the current stimulus in circuit simulation and in silicon, inducing a periodic step current onto the membrane. The examined neuron consists of two `DenMem` circuits with their membrane capacitances switched to 2 pF each.

Figure 6 shows results of some reproduced patterns according to [23] or [16] neighbored by their phase plane trajectory of $V$ and $V_{\text{adapt}}$. As the simulation describes an electronic circuit, the trajectories are continuous. All graphs have been recorded injecting a step current of 600 nA onto the membrane. $g_{\text{adapt}}$ and $g_l$ have been chosen equal in all simulations except tonic spiking to facilitate the nullclines:

$$V_{\text{adapt}} = -\frac{g_l}{a}\left(V - E_l\right) + \frac{g_l}{a}\Delta_T e^{\left(\frac{V - V_T}{\Delta_T}\right)} + E_l + \frac{I}{a} \qquad (9)$$

$$V_{\text{adapt}} = V; \qquad (10)$$

As described in [16], the AdEx model allows different types of *spike after potentials* (SAP). Sharp SAPs are reached if the reset after a spike sets the trajectory to a point, below the V-nullcline. If reset ends in a point, above the V-nullcline, the membrane voltage will be pulled down below the reset voltage $V_{\text{reset}}$ by the adaptation current.

The first pattern - tonic spiking with a sharp reset - can be reached by either setting $b$ to a small value and shrinking the adaptation time constant to make $V_{\text{adapt}}$ follow $V$ very fast - at least, the adaptation constant must be small enough to enable $V_{\text{adapt}}$ to regenerate $b$ in the inter-spike interval (ISI)- or by setting $a$ to zero. Here, a has been set to zero, while $g_l$ has been doubled to keep the total conductance at a similar level. Parameters between simulation and measurement are only roughly mapped, as the precise mapping algorithm is still in progress - on a real chip there is a variation of transistor parameters which still needs to be counterbalanced by parameter choice.

Spike-frequency adaptation is caused by enlarging $V_{\text{adapt}}$ at each detected spike, while still staying below the V-nullcline (equation 9). As metric, for adaptation [24] and [16] use the accommodation index:

$$A = \frac{1}{N-k-1}\sum_{i=k}^{N}\frac{\text{ISI}_i - \text{ISI}_{i-1}}{\text{ISI}_i + \text{ISI}_{i-1}} \qquad (11)$$

Here $k$ determines the number of ISI excluded from $A$ to exclude transient behavior [15, 24] and can be chosen as one fifth for small numbers of ISIs [24]. The metric calculates the average of the difference between two neighbored ISIs weighted by their sum, so it should be zero for ideal tonic spiking. For our results we get an accommodation index of $0 \pm 0.0003$ for fast spiking neurons in simulation and $-0.0004 \pm 0.001$ in measurement. For adaptation the values are $0.1256 \pm 0.0002$ and

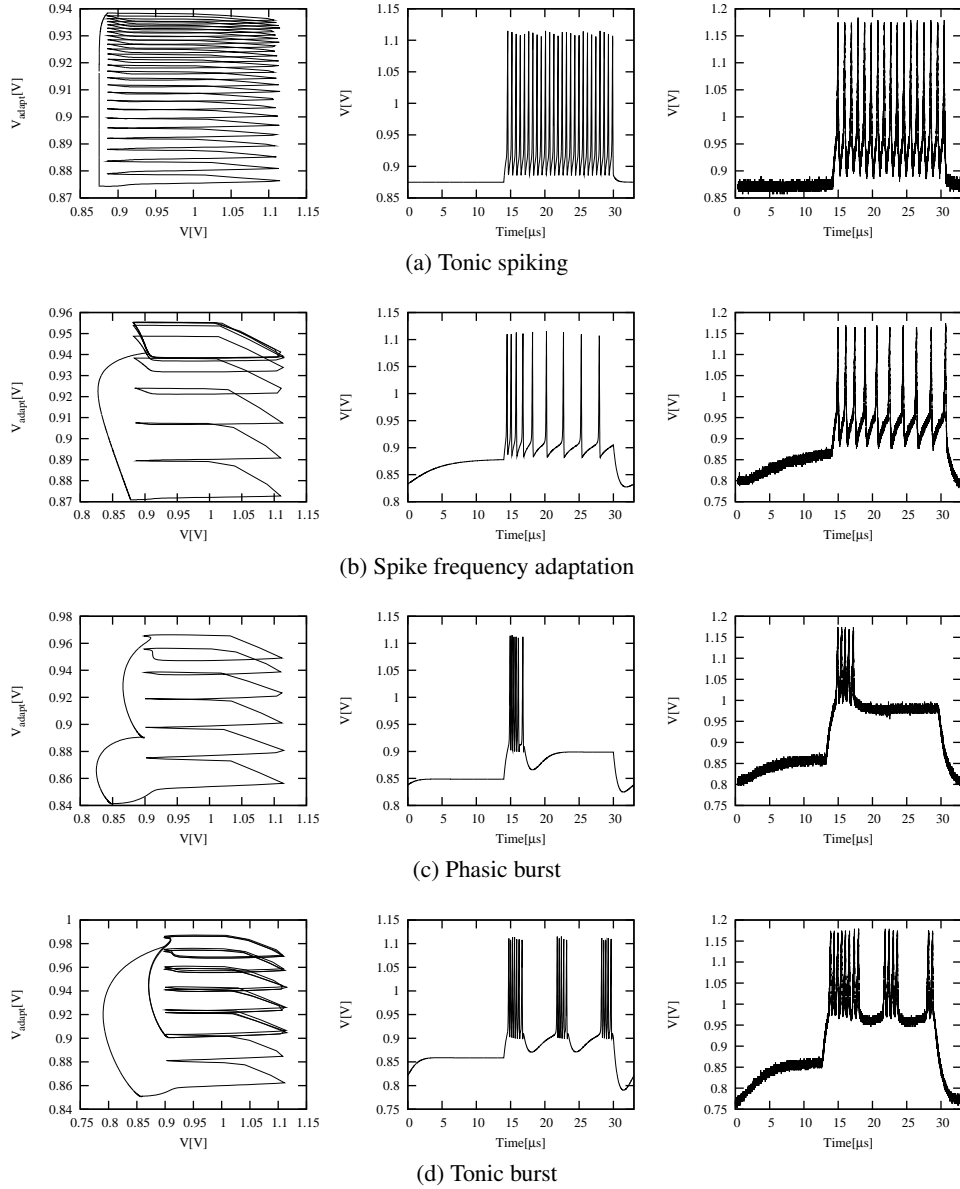

(a) Tonic spiking

(b) Spike frequency adaptation

(c) Phasic burst

(d) Tonic burst

Figure 6: Phase plane and transient plot from simulations and measurement results of the neuron stimulated by a step current of 600 nA.

$0.039 \pm 0.001$. As parameters have been chosen to reproduce the patterns obviously (adaptation is switched of for tonic spiking and strong for spike frequency adaptation) they are a little bit extreme in comparison to the calculated ones in [24] which are $0.0045 \pm 0.0023$ for fast spiking interneurons and $0.017 \pm 0.004$ for adapting neurons.

It is ambiguous to define a burst looking just at the spike frequency. We follow the definition used in [16] and define a burst as one or more sharp resets followed by a broad reset. The bursting results can be found in figure 6, too. To generate bursting behavior, the reset has to be set to a value above the exponential threshold so that $V$ is pulled upwards by the exponential directly after a spike.

As can be seen in figure 1, depending on the sharpness $\Delta_t$ of the exponential term, the exact reset voltage $V_r$ might be critical in bursting, when reseting above the exponential threshold and the nullcline is already steep at this point. The AdEx model is capable of irregular spiking in contrast to the Izhikevich neuron [25] which uses a quadratic term to simulate the rise at a spike. The

chaotic spiking capability of the AdEx model has been shown in [16]. In Hardware, we observe that it is common to reach regimes, where the exact number of spikes in a burst is not constant, thus the distance to the next spike or burst may differ in the next period. Another effect is that if the equilibrium potential - the potential, where the nullclines cross - is near $V_\mathrm{t}$, noise may cause the membrane to cross $V_\mathrm{t}$ and hence generate a spike (Compare phase planes in figure 6 c) and d) ).

Figure 6 shows tonic bursting and phasic bursting. In phasic bursting, the nullclines are still crossing in a stable fix point - the resting potential caused by adaptation, leakage and stimulus is below the firing threshold of the exponential.

Patterns reproduced in experiment and simulations but not shown here are phasic spiking and initial bursting.

# 4   Discussion

The main feature of our neuron is the capability of directly reproducing the AdEx model. It is neither optimized to be low power nor small in size in contrast to postulations by Livi in [6]. Nevertheless, it is low power in comparison to simulation on a supercomputer (estimated $100\,\mu\mathrm{W}$ in comparison to $370\,\mathrm{mW}$ on a Blue Gene/P [26] at an acceleration factor of $10^4$, computing time of Izhikevich neuron model [23] used as estimate.) and does not consume much chip area in comparison to the `synapse array` and communication infrastructure on the `HICANN` (figure 2). Complex individual parameterization allows adaptation onto different models. As our model is working on an accelerated time scale of up to $10^5$ times faster than biological real time, it is neither possible nor wanted to interact with systems relying on biological real time. Instead, by scaling the system up to about a million neurons, it will be possible to do experiments which have never been feasible so far due to the long duration of numerical simulations at this scale, i.e. allowing large parameter sweeps, dense real-world stimuli as well as many repetitions of experiments for gaining statistics.

Due to the design approach - implementing an established model instead of developing a new model fitting best to hardware devices - we gain a neuron allowing neuroscientist to do experiments without being a hardware specialist.

# 5   Outlook

The neuron topology - several `DenMems` are interconnected to form a neuron - is predestined to be enhanced to a multi-compartment model. This will be the next design step.

The simulations and measurements in this work qualitatively reproduce patterns observed in biology and reproduced by the AdEx model in [16]. A method to directly map the parameters of the AdEx quantitatively to the simulations has already been developed. This method needs to be enhanced to a mapping onto the real hardware, counterbalancing mismatch and accounting for limited parameter resolution.

Nested in the `FACETS` wafer-scale system, our neuron will complete the universality of the system by a versatile core for analog computation. Encapsulation of the parameter mapping into low level software and PyNN [12] integration of the system will allow computational neural scientists to do experiments on the hardware and compare them to simulations, or to do large experiments, currently not implementable in a simulation.

### Acknowledgments

This work is supported in part by the European Union under the grant no. IST-2005-15879 (FACETS).

## Footnotes

[1]Very large scale integration

[2]High Input Count Analog Neural Network

[3]metal-oxide-semiconductor field-effect transistor

### References

[1] Carver A. Mead and M. A. Mahowald. A silicon model of early visual processing. *Neural Networks*, 1(1):91–97, 1988.

[2] C. A. Mead. *Analog VLSI and Neural Systems*. Addison Wesley, Reading, MA, 1989.

[3] Misha Mahowald and Rodney Douglas. A silicon neuron. *Nature*, 354(6354):515–518, Dec 1991.

[4] E. Farquhar and P. Hasler. A bio-physically inspired silicon neuron. *Circuits and Systems I: Regular Papers, IEEE Transactions on*, 52(3):477 – 488, march 2005.

[5] J.V. Arthur and K. Boahen. Silicon neurons that inhibit to synchronize. In *Circuits and Systems, 2007. ISCAS 2007. IEEE International Symposium on*, pages 1186 –1186, 27-30 2007.

[6] P. Livi and G. Indiveri. A current-mode conductance-based silicon neuron for address-event neuromorphic systems. In *Circuits and Systems, 2009. ISCAS 2009. IEEE International Symposium on*, pages 2898 – 2901, 24-27 2009.

[7] Jayawan H.B. Wijekoon and Piotr Dudek. Compact silicon neuron circuit with spiking and bursting behaviour. *Neural Networks*, 21(2-3):524 – 534, 2008. Advances in Neural Networks Research: IJCNN '07, 2007 International Joint Conference on Neural Networks IJCNN '07.

[8] J. Schemmel, A. Grübl, K. Meier, and E. Muller. Implementing synaptic plasticity in a VLSI spiking neural network model. In *Proceedings of the 2006 International Joint Conference on Neural Networks (IJCNN)*. IEEE Press, 2006.

[9] J. Schemmel, D. Brüderle, K. Meier, and B. Ostendorf. Modeling synaptic plasticity within networks of highly accelerated I&F neurons. In *Proceedings of the 2007 IEEE International Symposium on Circuits and Systems (ISCAS)*, pages 3367–3370. IEEE Press, 2007.

[10] Alain Destexhe. Conductance-based integrate-and-fire models. *Neural Comput.*, 9(3):503–514, 1997.

[11] Daniel Brüderle, Eric Müller, Andrew Davison, Eilif Muller, Johannes Schemmel, and Karlheinz Meier. Establishing a novel modeling tool: A python-based interface for a neuromorphic hardware system. *Front. Neuroinform.*, 3(17), 2009.

[12] A. P. Davison, D. Brüderle, J. Eppler, J. Kremkow, E. Muller, D. Pecevski, L. Perrinet, and P. Yger. PyNN: a common interface for neuronal network simulators. *Front. Neuroinform.*, 2(11), 2008.

[13] R. Brette and W. Gerstner. Adaptive exponential integrate-and-fire model as an effective description of neuronal activity. *J. Neurophysiol.*, 94:3637 – 3642, 2005.

[14] FACETS. Fast Analog Computing with Emergent Transient States – project website. `http://www.facets-project.org`, 2010.

[15] Henry Markram, Maria Toledo-Rodriguez, Yun Wang, Anirudh Gupta, Gilad Silberberg, and Caizhi Wu. Interneurons of the neocortical inhibitory system. *Nat Rev Neurosci*, 5(10):793–807, Oct 2004.

[16] Richard Naud, Nicolas Marcille, Claudia Clopath, and Wulfram Gerstner. Firing patterns in the adaptive exponential integrate-and-fire model. *Biological Cybernetics*, 99(4):335–347, Nov 2008.

[17] J. Schemmel, J. Fieres, and K. Meier. Wafer-scale integration of analog neural networks. In *Proceedings of the 2008 International Joint Conference on Neural Networks (IJCNN)*, 2008.

[18] J. Fieres, J. Schemmel, and K. Meier. Realizing biological spiking network models in a configurable wafer-scale hardware system. In *Proceedings of the 2008 International Joint Conference on Neural Networks (IJCNN)*, 2008.

[19] J. Schemmel, D. Brüderle, A. Grübl, M. Hock, K. Meier, and S. Millner. A wafer-scale neuromorphic hardware system for large-scale neural modeling. In *Proceedings of the 2010 IEEE International Symposium on Circuits and Systems (ISCAS)*, pages 1947–1950, 2010.

[20] T.S. Lande, H. Ranjbar, M. Ismail, and Y. Berg. An analog floating-gate memory in a standard digital technology. In *Microelectronics for Neural Networks, 1996., Proceedings of Fifth International Conference on*, pages 271 –276, 12-14 1996.

[21] Alain Destexhe, Diego Contreras, and Mircea Steriade. Mechanisms underlying the synchronizing action of corticothalamic feedback through inhibition of thalamic relay cells. *Journal of Neurophysiology*, 79:999–1016, 1998.

[22] Martin Pospischil, Maria Toledo-Rodriguez, Cyril Monier, Zuzanna Piwkowska, Thierry Bal, Yves Frégnac, Henry Markram, and Alain Destexhe. Minimal hodgkin–huxley type models for different classes of cortical and thalamic neurons. *Biological Cybernetics*, 99(4):427–441, Nov 2008.

[23] Eugene M. Izhikevich. Which Model to Use for Cortical Spiking Neurons? *IEEE Transactions on Neural Networks*, 15:1063–1070, 2004.

[24] Shaul Druckmann, Yoav Banitt, Albert Gidon, Felix Schrmann, Henry Markram, and Idan Segev. A novel multiple objective optimization framework for constraining conductance-based neuron models by experimental data. *Front Neurosci*, 1(1):7–18, Nov 2007.

[25] Eugene M. Izhikevich. Simple Model of Spiking Neurons. *IEEE Transactions on Neural Networks*, 14:1569–1572, 2003.

[26] IBM. System blue gene solution. `ibm.com/systems/deepcomputing/bluegene/`, 2010.

